# FineStyle: Fine-grained Controllable Style Personalization for Text-to-image Models

**Gong Zhang**[1,2]* **Kihyuk Sohn**[3]† **Meera Hahn**[2] **Humphrey Shi**[1] **Irfan Essa**[1,2]

[1]Georgia Tech  [2]Google DeepMind  [3]Meta Reality Labs

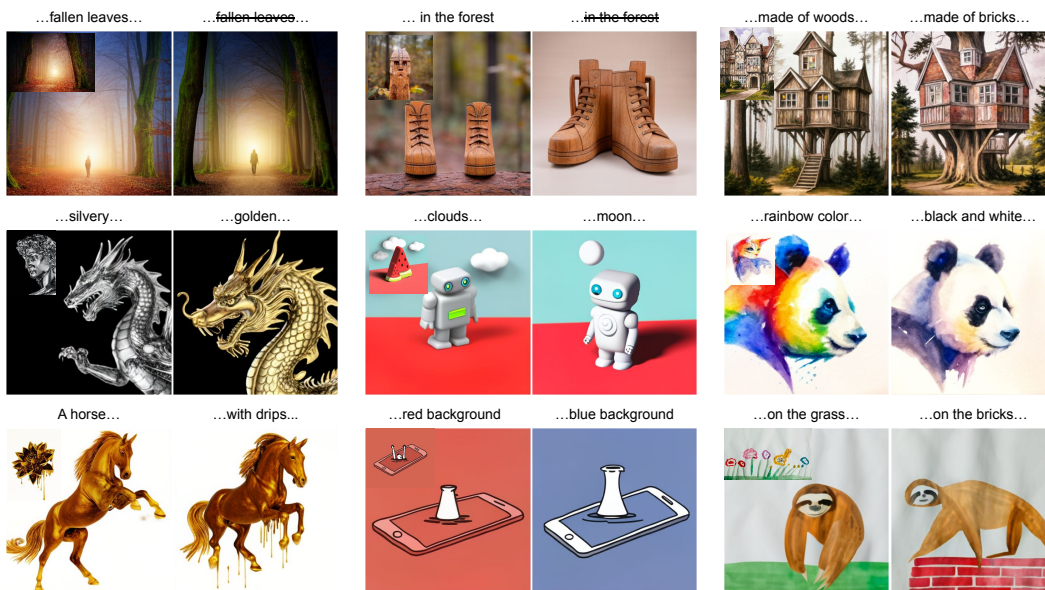

Figure 1: Demonstration of fine-grained style controllability of FineStyle. Nine image pairs are generated by personalized text-to-image models, each of which is fine-tuned on a respective, single style reference image displayed at the corner of the left image of each pair. Fine-grained concepts are written on top of the images for comparisons, showing the nuanced compositionality encompassing color, foreground object, background, and textures. Full prompts are available in Appendix A.1. Visit https://github.com/SHI-Labs/FineStyle for code and more examples.

## Abstract

Few-shot fine-tuning of text-to-image (T2I) generation models enables people to create unique images in their own style using natural languages without requiring extensive prompt engineering. However, fine-tuning with only a handful, as little as one, of image-text paired data prevents fine-grained control of style attributes at generation. In this paper, we present FineStyle, a few-shot fine-tuning method that allows enhanced controllability for style personalized text-to-image generation. To overcome the lack of training data for fine-tuning, we propose a novel concept-oriented data scaling that amplifies the number of image-text pair, each of which focuses on different concepts (e.g., objects) in the style reference image. We also identify the benefit of parameter-efficient adapter tuning of key and value kernels of

cross-attention layers. Extensive experiments show the effectiveness of FineStyle at following fine-grained text prompts and delivering visual quality faithful to the specified style, measured by CLIP scores and human raters.

# 1 Introduction

Text-to-image (T2I) models [38, 4, 40] have become a powerhouse driving various modern image-creation applications [11, 28, 8, 34] to generate unique artworks of diverse styles from natural language prompts. However, it is often challenging to faithfully describe the visual look of a style in pure text form. To better leverage the generation capability of these models, a series of works [39, 41, 10, 27] extend the one-step text-to-image generation paradigm to few-shot fine-tuning with a set of images followed by a personalized text-to-image generation. Following this two-step paradigm, users can create novel images inheriting the visuals of reference images without extensive prompt engineering [45]. Although these works and their successors [52, 48] have made the text-to-image paradigms capable of conveniently generating with the guidance of reference images, they often result in content leakage, where visual clues of unwanted contents from the reference image appear in generated images. For example, as shown in Fig. 2, StyleDrop [41], state-of-the-art method for one-shot style-tuning of text-to-image generation model, suffers from a content leakage of generating spindle leaves in the background when asked to generate a sneaker.

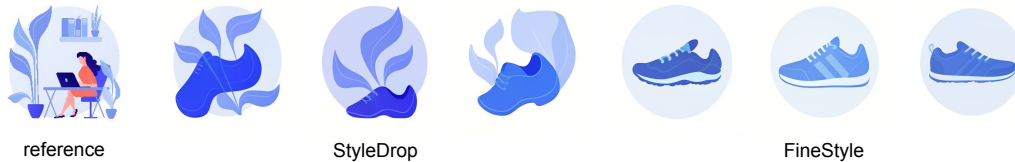

reference            StyleDrop            FineStyle

Figure 2: StyleDrop [41] tends to leak contents of the style reference image into generated images, such as the spindle leaves in the background of "a sneaker", even though it is not included in the text prompt. FineStyle learns by pinpointing desirable style attributes (e.g., flat cartoon vector art) and mitigates the leakage of unwanted content (e.g., spindle leaves) at generation.

Why does the content leakage happen for few-shot or one-shot fine-tuning? When training text-to-image generation models on a large amount of image-text paired data covering a wide variety of visual concepts, it is easy to decompose individual concepts both in the text and visual spaces and associate those concepts between these two spaces. On the other hand, it is challenging to correctly associate visual concepts with the corresponding text phrases using only a few or one training images, as the contents in the image are highly entangled. As such, [41] has proposed an iterative fine-tuning strategy where later rounds of fine-tuning are done with a set of generated synthetic images curated by automated or human feedback. This synthetic fine-tuning on many images is shown to be effective at disentangling visual concepts of subject and style, leading to an improved image-text association for personalized text-to-image generation. However, iterative training is prohibitive as it requires more computing and human resources.

This paper aims to develop an efficient fine-tuning method for T2I models from a single reference image. The primary insight is to fine-tune the models with additional objectives focusing on multiple fine-grained concepts within a text-image pair rather than solely on the text-image pair itself. To this end, we decompose a single style reference image and its text prompt into multiple concept-oriented sub-image-text pairs, as in Fig. 3(b–d). Concretely, the sub-image-text pair describes an individual fine-grained concept of a style reference image, such as a laptop, woman, or plant, instead of the original text prompt describing an entire image. Users may identify as many concepts of interest as they want from a main text prompt. Therefore, the more comprehensive the prompt is, the better the concept grounding our method can provide. We leverage the disentanglement capability residing in cross-attention layers in pretrained T2I models to obtain the spatial location of individual concepts.

In addition to the concept-oriented data scaling, we extend parameter-efficient fine-tuning to directly modify weights in the cross-attention layer instead of adding residuals to hidden features via adapters [18, 41]. The intuition behind this strategy is as follows: first, fine-tuning is a task that adds new cross-modal knowledge of a text-image pair to T2I models, while cross-attention layers are

places where the kind of knowledge is stored; as such, adapters at weight matrices of cross-attention layer has the potential to bring better expressivity than using adapters to transform hidden features outside cross-attention layer.

We test our method on Muse [4] as the T2I backbone over a diverse set of style reference images. We show that our method, FineStyle, is able to generate style-consistent images from text prompts while mitigating the content leakage, as in Fig. 2. Furthermore, our fine-tuning method promotes concept disentanglement, enabling novel applications like style editing. This empowers users to modify granular style attributes of a reference image, as visualized in Fig. 7. Extensive evaluation measuring the semantic and style fidelity using CLIP [35] and user studies show the enhanced performance of our method compared to baselines.

## 2 Related Work

**Text-to-image Synthesis**. Compared to unconditional image synthesis [21, 17, 42], in which models learn to create images randomly resembling their training data distribution, conditional image synthesis [37, 31, 40, 50, 4, 46, 47, 12] introduces extra conditioning on text/image prompts to guide the generative process. This explicit conditioning has made a series of downstream image-generative tasks more controllable with natural language prompts such as text-guided inpainting [44] and image editing [2, 22, 52].

**Cross-attention mechanism in T2I** has been used to implement the interaction between visual and textual features, enabling text conditioning in image synthesis. [32] demonstrates that cross-attention maps align the concepts in text prompts with their corresponding spatial positions on the generated image. A few works [13, 5, 51] have confirmed that modifying attention weights affects the layout and content in resulting synthesis.

**T2I Personalization Synthesis** extends the capability of pre-trained T2I models to generate images of novel concepts outside their training set. Given a small collection of reference images about a concept, it works by either optimizing the T2I model itself [39, 10, 22, 1] or injecting extracted image features into the generative process [14, 30, 48]. Dreambooth [39] fine-tunes all parameters of a T2I model and gains decent fidelity in synthesizing the target concept, but it comes at the cost of training and storage efficiency. Adopting the idea of parameter-efficient fine-tuning (PEFT) from NLP [18, 20, 26, 6], StyleDrop [41] learns a set of lightweight adapter layers appended to each transformer block of a generative vision transformer [4] from a single style image to improve data and training efficiency. On the other hand, StyleAligned [14] can generate a consistent image set of a style by extending the self-attention at inference time to encompass features of a reference image. As such, it achieves style personalization without optimization. However, they all implicitly deem a style image and its text description indivisible and ignore the importance of aligning fine-grained style and description in the context of more controllable personalization synthesis.

## 3 Preliminary

In this section, we review Muse [4], a masked generative transformer for text-to-image generation, and StyleDrop [41], a few-shot style-tuning built on Muse for style-personalized text-to-image generation.

**Muse** [4] is a masked generative image transformer, or MaskGIT [3]. It contains a pre-trained text encoder $\mathtt{T}$, an image encoder $\mathtt{E}$, a decoder $\mathtt{D}$, and a generative transformer $\mathtt{G}$. Muse uses T5-XXL [36] for $\mathtt{T}$ and VQGAN [49, 7] for $\mathtt{E}$ and $\mathtt{D}$. $\mathtt{E}$ encodes an image from pixel space to a sequence of discrete visual tokens $v \in \mathcal{E}$ while $\mathtt{T}$ encodes a text prompt into textual token space $\mathcal{T}$. Namely, we are interested in obtaining $\mathtt{G} : \mathcal{E} \times \mathcal{T} \to \mathcal{L}$ that takes in visual and textual tokens and outputs logits $\in \mathcal{L}$. $\mathtt{G}$ is trained to reconstruct masked visual tokens with conditioning textual tokens from a large text-image pair dataset $\mathcal{D}$ [40].

$$L = \mathbb{E}_{(x,t)\sim\mathcal{D}, m\sim\mathcal{M}} \left[ \mathtt{CE}(\mathtt{E}(x), \mathtt{G}(\mathcal{M}(\mathtt{E}(x), m), \mathtt{T}(t))) \right] \qquad (1)$$

where $(x, t)$ is an image-text pair and $\mathcal{M}$ is a uniformly distributed mask sampling strategy with a mask ratio as a coefficient. $\mathtt{CE}$ is a weighted cross-entropy loss calculated by summing over losses at masked visual tokens. Once $\mathtt{G}$ is trained, an image $\mathcal{I}$ is synthesized by iterative decoding [4, 3] visual logits given a text prompt and an initial sequence of visual tokens. A sampling strategy $\mathtt{S}$ samples

visual tokens from output logits. Finally, D maps visual tokens at the last step to the image of pixels.

$$\mathcal{I} = \texttt{D}(v_K), v_k = \texttt{S}(\texttt{G}(v_{k-1}, \texttt{T}(t)) + \lambda(\texttt{G}(v_{k-1}, \texttt{T}(t)) - \texttt{G}(v_{k-1}, \texttt{T}(n)))) \qquad (2)$$

where $k \in [1, K]$ is the sampling step, $t$ is the text prompt and $n$ is the null prompt. The term with $\lambda$ as a coefficient is classifier-free guidance [16].

With the cascade design from low to high resolution, Muse contains several sub-modules: a pair of low-res and high-res VQGAN operating at $256 \times 256$ and $512 \times 512$ resolutions, respectively, a base transformer for decoding low-res image tokens and a super-resolution transformer for translating low-res image tokens to high-res image tokens. We refer readers to [4] for additional details on the Muse model configurations.

**StyleDrop** [41] is a few-shot style personalized text-to-image generation model built on the Muse [3]. Given a dataset $\mathcal{D} = \{(x_i, t_i)\}_{i=1}^{N}$ of $N$ image-text pairs and a generative model G, we are interested in obtaining $\hat{\texttt{G}} : \mathcal{E} \times \mathcal{T} \times \Theta \to \mathcal{L}$ that takes in an extra set of trainable parameters $\theta \in \Theta$ (i.e., adapter tuning [18]) to generate logits of visual tokens. Now the cross entropy loss over $\theta$ becomes:

$$L_\theta = \mathbb{E}_{(x,t) \sim \mathcal{D}, m \sim \mathcal{M}} \left[ \texttt{CE}(\texttt{E}(x), \texttt{G}(\mathcal{M}(\texttt{E}(x), m), \texttt{T}(t), \theta)) \right] \qquad (3)$$

With $\hat{\texttt{G}}$, users can generate novel images with style descriptor prompts that follow the style represented by $\mathcal{D}$. In practice, the number of image-text pairs in $\mathcal{D}$ could be very small, resulting in few-shot or even one-shot fine-tuning.

## 4 Method

Similarly to the StyleDrop [41], we build the FineStyle on Muse [4] via fine-tuning. In this section, we first discuss the challenges of fine-grained concept alignment in few-shot fine-tuning (Sec. 4.1). Then, we introduce building blocks of the FineStyle, a concept-oriented data scaling (Sec. 4.2) and parameter-efficient adapters (Sec. 4.3). Fig. 3 provides an overview of our method.

### 4.1 Challenges of Fine-grained Concept Alignment in Few-shot Fine-Tuning

Recent text-to-image models demonstrate a strong image generation capability from natural language prompts [31, 40, 50, 4]. Although these models are trained to generate images from text prompts that describe an image as a whole, fine-grained concept alignment capabilities emerge through training on a large volume of image-text pairs. This enables models to compose multiple fine-grained concepts to generate a cohesive image semantically. However, it remains a challenge to achieve such a fine-grained concept alignment in few-shot fine-tuning, as the number of training examples are too limited to learn associations between textual concept descriptions and their visual representations.

Such an issue has been identified in a previous work [41]. As a result of a poor concept disentanglement in few-shot fine-tuning, a content leakage happens at generation, i.e., some visual concepts are unexpectedly generated in an image even though they are not included in the text prompt. See Fig. 2 as an example, where generated images on the left by StyleDrop contain spindle leaves in the background, though the model is asked to generate an image of a sneaker. To address this, iterative training with either automated or human feedback [41] is proposed, where additional rounds of fine-tuning are conducted on synthetic images generated by models from earlier iterations. While this approach has proven effective, it suffers from expensive labor costs, extra annotation time, dependency on the quality of synthetic images, and risk of performance deteriorating due to human selection bias.

### 4.2 Concept-oriented Data Scaling for Masked Decoding

As opposed to an iterative training of [41], we seek for an efficient, single-stage data scaling approach to enhance the concept alignment over target fine-tuning domains with limited data. This suggests us to explore ways to leverage the concept composition within limited data.

One critical observation is that the style reference image often contains multiple concepts. Many are spatially decomposable concepts (e.g., foreground subjects and background scenes and objects) while sharing the consistent visual style. For example, as in Fig. 3, a style reference image (top-left)

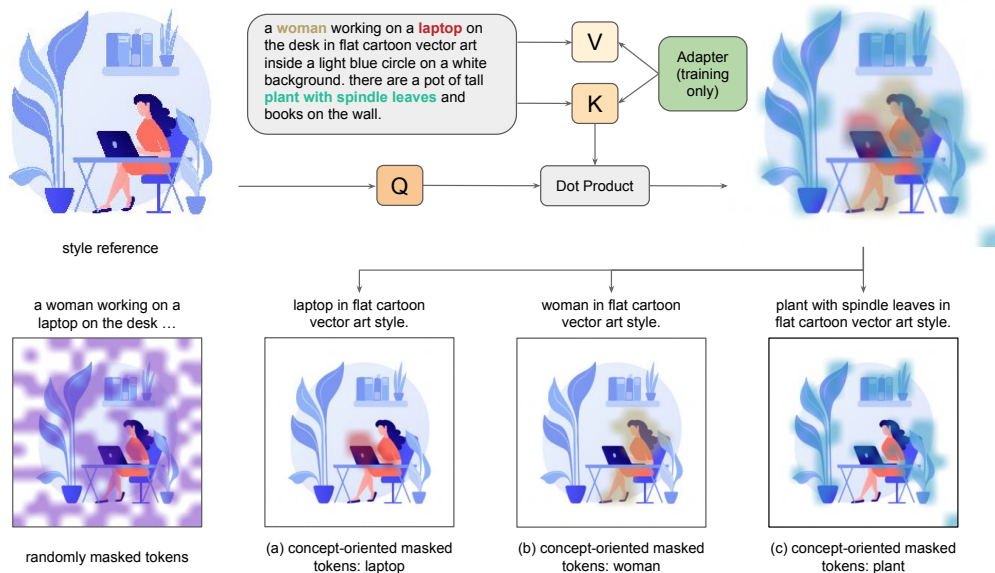

Figure 3: An overview of FineStyle framework, including the concept-oriented data scaling workflow (arrows) and PEFT adapters (green box) applied to key and value kernels within transformer blocks. **The workflow** starts from the top-left with a single image-text pair containing user-specified concepts in colored texts. The image-text cross-attention map (top-right) is retrieved from the dot product between the query and key matrices. From an attention map, we aggregate attention values corresponding to the user-specified concepts (e.g., laptop, woman, or plant) to create extra training pairs, as in (a), (b), and (c), each of which focuses on different subjects, derived from a single style image.

contains multiple concepts, including a woman, a laptop, a plant, and a bookshelf. Furthermore, the objective of fine-tuning is to learn these implicit alignments between these visual and textual concepts, e.g., word "woman" to visual woman, word "laptop" to visual laptop on the desk, word "a pot of tall plant with spindle leaves" to visual a pot of plant, "words "flat cartoon vector art" to visuals of all concepts in the image. While we wish such a decomposition of an image and the text prompt and an association between individual textual and visual concepts naturally occur, it is not designed to do so. Taking these observations into consideration, we propose learning through concept-oriented conditioning on dedicated captions to enhance concept alignment.

**Concept Decomposition.** To achieve this, we first decompose a text prompt into multiple concept-oriented sub-text prompts. Similarly to [41], we construct a text prompt for a style reference image by combining the subject and style phrases. Specifically, we build a comprehensive text prompt [25] to describe multiple subjects, styles, and background attributes in the image. For example, as in Fig. 3, we use "woman", "laptop", "a pot of plant with spindle leaves", and "bookshelf" for foreground subjects and "flat cartoon vector art", "a light blue circle", and "white background" for style and background attributes. Then, we create a few sub-prompts by combining a prominent concept and style phrases from the text prompt, e.g., "{concept phrase} in {style phrase} style". As a result, in addition to the original style reference image and the text pair (Fig. 3(a)), we get a couple more text prompts such as "a laptop in flat cartoon vector art style" as in Fig. 3(b) or "a plant with spindle leaves in flat cartoon vector art style" as in Fig. 3(d), each of which focuses on a different subject in a style reference image. The process could be done manually or automated by using state-of-the-art vision large language models (vLLMs), as shown in Appendix A.6.

**Training with Concept-oriented Masking.** One way to train a model with decomposed concepts is to create a set of sub-images (and their corresponding sub-prompt pairs) by cropping around the concept area. While this sounds straightforward, it will introduce extra cumbersome such as mismatched image ratio. Instead of explicitly cropping an image with bounding boxes, we propose concept-oriented masking, which replaces the traditional strategy of randomly and uniformly masking visual tokens across the entire image with targeted masking of concept-specific areas. Note that

the Muse model is trained to predict masked tokens, where the masked token is chosen uniformly at random as in Fig. 3(a). We construct the concept-oriented mask from the segmentation map that aggregates the cross-attention weights of the pretrained Muse model for the corresponding concept, and tokens inside the concept-oriented mask serve as prediction targets. Details on deriving a segmentation map from aggregated cross-attention weights are provided in Appendix A.5. Fig. 3(a–c) shows the concept-oriented masks with corresponding sub-prompts. During the training, all four examples have an equal chance of appearing in a batch.

## 4.3 Parameter-Efficient Adapter for Masked Generative Image Transformer

Parameter-efficient adapters [20, 33, 6, 9] have become the new norm for fine-tuning a large model. Compared to fine-tuning the entire model, the adapters have the advantage of being small and easily interchangeable. Most existing works have tested adding adapters at various places in T2I models, such as token embeddings [10] and intermediate hidden features [41, 30]. We argue that adding adapters to cross-attention layers is more beneficial for fine-grained style personalization. First, fine-grained style controllability with text prompts depends on the precise alignment between visual and textual features, and cross-attention layers are the places where this cross-modal interaction happens. Second, in the T2I generative model, we define the hidden features as inputs and outputs to self-attention, cross-attention or MLP layers in a generative transformer. They are usually of shape in [batch, num_visual_token, feat_dim] and finally used to predict logits of visual tokens, thus containing substantial neighborhood and spatial information. Overall, we limit our adapter to key and value kernels corresponding to textual prompts at the cross-attention layer, leaving the query kernel untouched.

Unlike the typical application of a dedicated LoRA layer to each transformer block of the image decoder, our method employs a singular main LoRA layer but modifies it with distinct biases for each transformer block. This adaptation reduces the number of trainable parameters and aims to mitigate potential overfitting issues, a critical aspect in maintaining model generalizability.

**Sampling with Adapter.** With adapted transformer $\hat{\mathtt{G}}$, visual tokens $v_k$ is obtained as below:

$$v_k = \hat{\mathtt{G}}(v_{k-1}, \mathtt{T}(t)) + \lambda_1(\hat{\mathtt{G}}(v_{k-1}, \mathtt{T}(t)) - \mathtt{G}(v_{k-1}, \mathtt{T}(t))) + \lambda_2(\hat{\mathtt{G}}(v_{k-1}, \mathtt{T}(t)) - \hat{\mathtt{G}}(v_{k-1}, \mathtt{T}(n))) \quad (4)$$

Compared to Eq. 2, we have an extra term with $\lambda_1$, which computes the logit residuals of prompt $t$ between adapted model and original model. Therefore, $\lambda_1$ controls the strength of style, while the term with $\lambda_2$ is classifier-free guidance for adapted model.

# 5 Experiment

We adopt the evaluation set from [41] containing 24 styles encompassing fine-art oil painting, 3D rendering, and sculpture. In Sec. 5.2, we report qualitative results of FineStyle and novel applications brought by enhanced fine-grained concept alignment. In Sec. 5.3, we test the semantic and style consistency of FineStyle-generated images using the CLIP score and human evaluation. In Sec. 5.4, we conduct ablations on components of our method. Implementation details are in Appendix A.3.

## 5.1 Evaluation Setup

As the style example in Fig. 3, we define style descriptor ("flat cartoon vector art") and unique fine-grained style properties (e.g., "inside a light circle" and "white background"). During the evaluation, we create simple prompts in the pattern of "{subject} in {style descriptor} style" for synthesis unless otherwise specified.

## 5.2 Qualitative Results

Fig. 1 shows the robust fine-grained style controllability using our method by modifying those unique style properties in training prompts. The prompts used for generation are composed of a subject, style descriptor, and style properties. All the comparison pairs are generated using the same random seed with only one property being different. It is clear from the result that FineStyle supports the control over properties such as color, texture, background, and decoration of a subject.

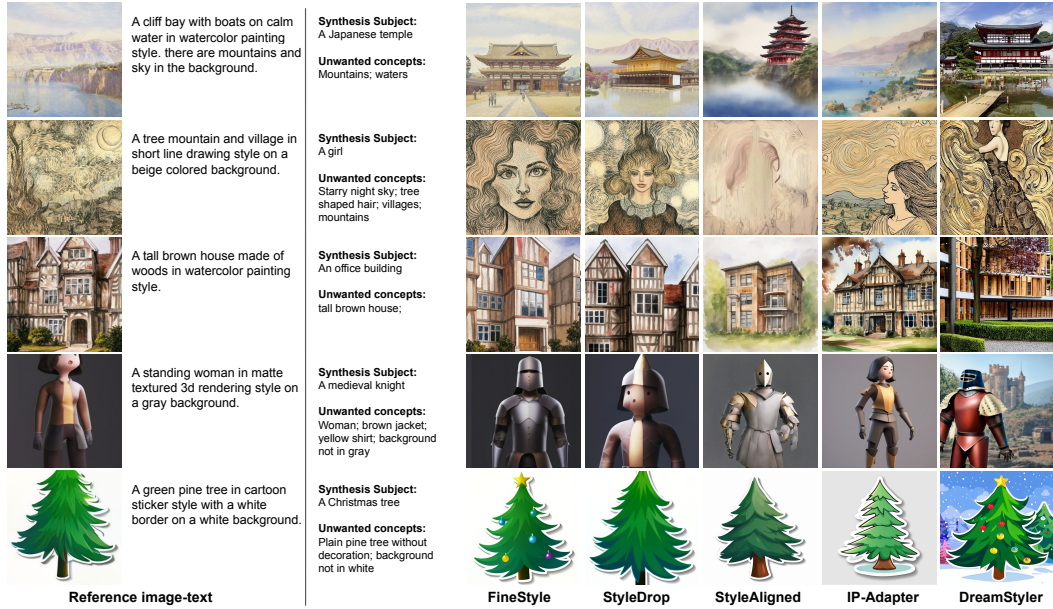

Figure 4: Qualitative comparison between FineStyle and various baselines. Unwanted concepts list those appearing in training prompts but should not be in synthesis prompts.

Fig. 4 compares FineStyle with baselines [15, 1, 48, 41] in 5 styles. The same prompt is used for both models to generate a set of 2 images in one inference pass. We can see that FineStyle consistently outperforms baselines regarding fine-grained concept alignment. The first example shows the content leakage problem: the bay and mountains entangle with "watercolor painting", causing them to leak into the generation of "a Japanese temple". Furthermore, StyleAligned [15] almost replicates the layout of the bay and cliffs from the reference image. In the second and third rows of results generated by StyleDrop [41], the starry night sky and the tree-shaped hair keep appearing in the generations of "a girl" and "modern office building" deteriorates to almost the same "house" in the reference image.

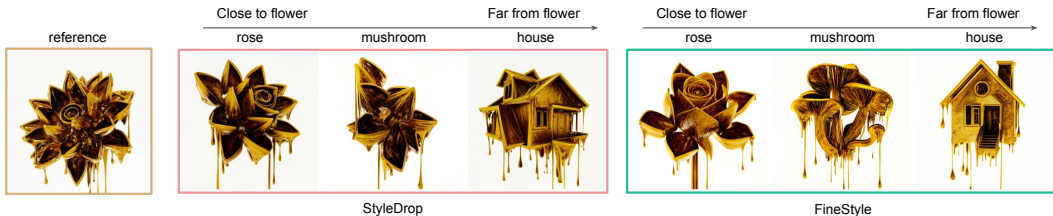

Figure 5: Generated images of "melting golden 3d rendering" style from text prompts of subjects whose semantic distance to the reference subject ("flower") is gradually changed from close to far. StyleDrop creates images that follow the text prompt when the subject is far from the reference subject. In contrast, FineStyle creates images of subjects even when they are semantically close ("rose" or "mushroom") to the reference subject.

This phenomenon highly correlates to the semantic distance between the training example and the generation prompt. To better understand this phenomenon, we use a series of concepts ranging from semantic closeness to farness to construct comparison prompts. In Fig. 5, we test it using the style "melting golden 3d rendering". As the training image is a flower with triangle-shaped petals, we choose the semantic axis to be a flower specie, a plant, and a building. From left to right, the generations of StyleDrop are improving from flower-shaped objects to houses, even though some triangles can still be seen in certain parts of the house. On the other hand, FineStyle performs better all along the semantic axis with desirable overall style consistency. This suggests that enhancing fine-grained concept alignment can effectively counter the phenomenon.

### 5.2.1 Extensive Style Control

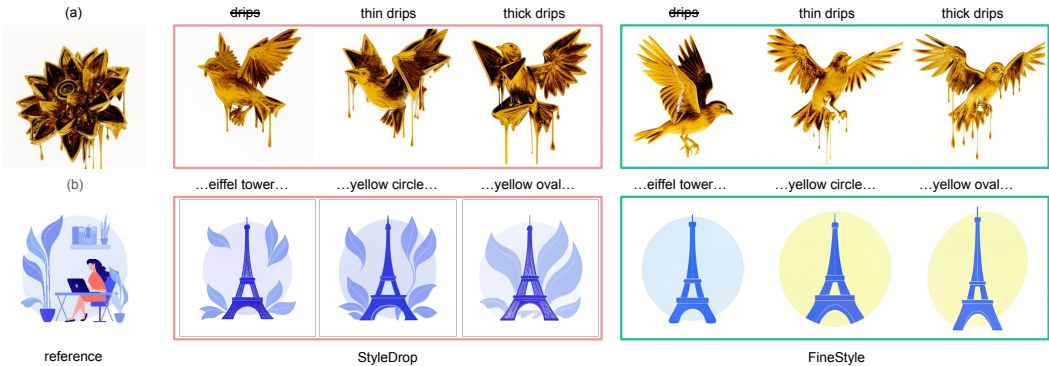

Figure 6: Extensive Style Control. (a) modifies a fine-grained style by omitting it or adding decoration to it. (b) controls multiple fine-grained styles at the same time.

Due to its fine-grained concept alignment, FineStyle allows extensive control over specific style attributes, even with limited visual representation. In Fig. 6(a), we demonstrate style control over melting drips. We adjust the state of the drips by omitting certain words or adding decorative elements. Results within the red box differ from those in the green box, showing no substantial changes in drip thickness or absence. Typically, without precise concept alignment, fine-tuned adapters tend to focus on large-area concepts, as seen with the spindle leaves in Fig. 6(b).

Furthermore, Fig. 6(b) demonstrates the feasibility of controlling two style attributes at the same time, suggesting the style adapter obtained from our fine-tuning algorithm is more compatible with the compositionality learned by the base model rather than unquestioningly learning to reconstruct every detail of training image at the same time.

### 5.2.2 Controllable Reference Image Variation

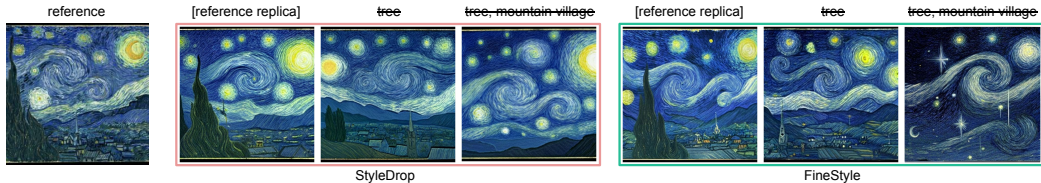

Figure 7: An example reference image variation. The last image without "tree, mountain, village" is synthesized with the prompt "a clear starry night sky close up in oil painting style on a blue background".

Given an image and a conditioning mechanism, generative models can generate image variations. Usually, these variations happen only at the granular style property level with a relatively similar image structure. We note that during one-shot fine-fintuning, the training image can be faithfully reconstructed with more training steps. While this might result in style overfitting, it opens the possibility of an interesting application of controllable image variation. As in Fig. 7, both StyleDrop and FineStyle can reconstruct the training image with minimum fidelity loss. However, only FineStyle can get a clean, starry night variation without the traces of trees, mountains, and black borders.

### 5.3 Quantitative Results

We synthesize images by combining a filtered version of Parti [50] prompts and 10 styles from the evaluation set, details in Appendix A.2. There are 190 examples in Parti prompts. Each one describes a composition of a subject. The subject comes with its superclass to reduce semantic ambiguity (e.g. A cat, animals, in watercolor painting style.). We generate 4 images for each example, adding to 760 images for a style.

**CLIP score.** We utilize CLIP [35] to calculate `Text` (text-image) and `Style` (image-image) scores. The `Text` score is between a generated image and its text prompt, measuring how well the image follows it. The `Style` score is between generated and style reference images, measuring style fidelity. However, it is not the higher the score, the better since high scores might indicate content leakage or mode collapsing.

Table 1: CLIP scores measuring image-text similarity (`Text`) and image-image similarity (`Style`). We test FineStyle alongside two variants: (a) with data scaling and a feature adapter after transformer layers, and (b) without data scaling, using an adapter at key and value kernels within transformer layers. FineStyle demonstrates the best balance between text and style scores.

| Method | data scaling | adapter | Text score ($\uparrow$) | Style score ($\uparrow^\Upsilon$) |
|---|---|---|---|---|
| Muse | - | - | 0.320 | 0.552 |
| StyleDrop | $\times$ | feature | 0.297 | 0.708 |
| variant (a) | $\checkmark$ | feature | 0.296 | 0.730 |
| variant (b) | $\times$ | kv kernel | 0.308 | 0.686 |
| FineStyle | $\checkmark$ | kv kernel | 0.314 | 0.661 |

FineStyle achieves higher `Text` scores than StyleDrop (0.314 v.s. 0.297) and reasonable `Style` scores (0.661, higher than Muse's 0.552 and lower than StyleDrop's 0.708). Since the content leakage problem results in prompt-image misalignment, the improved `Text` scores imply that FineStyle alleviates the problem while still maintaining competitive style fidelity.

**Human Evaluation.** Given a reference image and a pair of synthesized images from two comparable models, users are asked to select the one that 1) corresponds better to the style of the reference image (`Style`); 2) better matches the prompt (`Text`); 3) makes more common sense. For example, given a prompt of "a goat with drips in melting golden 3d rendering style on a solid white background", a generated image, as in Fig. 12, should show a goat with 4 legs as it aligns with the common understanding, even though it is not specified in the prompt. We provide more details on the human evaluation in the Appendix A.4.

Table 2: We report the scores of human evaluation over pairs of generated images from StyleDrop and FineStyle. Images generated from FineStyle are much preferred by users regarding image-text alignment and structure / common sense alignment.

| | StyleDrop | tie | FineStyle |
|---|---|---|---|
| Text ($\uparrow$) | 10.8% | 21.9% | 67.1% |
| Style ($\uparrow$) | 43.5% | 27.5% | 28.9% |
| Structure / Common Sense ($\uparrow$) | 23.6% | 11.1% | 65.2% |

The results in Tab. 2 demonstrate that FineStyle is significantly preferred in `Text` and Structure/Common Sense. While StyleDrop wins in `Style`, there is still a large portion of ties, showing FineStyle maintains comparable or better performance for more than half (56.4%) of the test cases. These results are congruent with the CLIP score evaluations in Tab. 1, but additionally provide information about common sense reasoning.

## 5.4 Ablation Study

### 5.4.1 Concept-oriented data scaling and KV adapter

We study the effectiveness of the main components in FineStyle by training model variants that disable one of them. The full FineStyle uses concept-oriented data scaling (data scaling) and adapters at key and value kernels (kv). In contrast, StyleDrop uses adapters at hidden features (feat) after a transformer layer. We train two variants: (a) data scaling with feat adapter and (b) only kv adapter. Comparing (a) with (b) in Tab. 1, we see that (a) has a lower `Text` score (0.296) but a problematically

---

$^\Upsilon$High style scores may suggest potential overfitting to the style image since they measure the similarity between generated images and style images.

high `Style` score (0.730). These scores align with our expectation that feat adapter tends to capture visual and spatial information that may ignore dynamic style compositions, leading to a problematic high `Style` score but worse controllability. Moreover, (b) gets a better `Text` score (0.308) without data scaling, which indicates the kv adapter is a better design for disentangling fine-grained styles in a few-shot setting.

### 5.4.2 Inference Hyperparameters

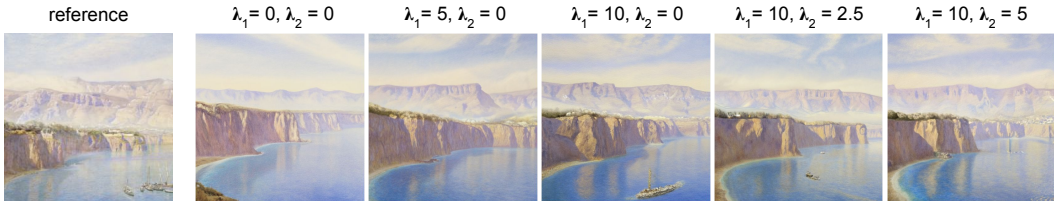

Figure 8: Effects of style ($\lambda_1$) and semantic ($\lambda_2$) guidance scales. The text prompt at training is "a cliff bay with boats on calm water...", while the text prompt at generation does not include "with boats on calm water".

We present the inference formula in Eq. 4. It has two hyperparameters $\lambda_1$ and $\lambda_2$ controlling style and semantics, respectively. In Fig. 8, we test their effects by changing them. As $\lambda_1$ increases, the synthesized image obtains more details from the reference. At $\lambda_1 = 10$, boats appear on the water, even though they are not in the prompt. This signifies that learned style unfavorably dominates the sampling and causes content (e.g., boats) to leak. Then, we increase $\lambda_2$, and the boats gradually vanish, making the image following the prompt more accurate.

## 6  Conclusion

In this paper, we introduce FineStyle, a method for style personalization of text-to-image models that requires only a single reference image. It comprises two key components: concept-oriented data scaling and an adapter applied to key and value kernels. Central to our approach is the utilization of the cross-attention mechanisms inherent in pre-trained text-to-image models. By leveraging this existing cross-modal knowledge, FineStyle effectively tailors style elements with precision, allowing for detailed customization in generated images.

## Acknowledgement

We use images collected in [41] for experiments. Image sources and attribution of ownership is provided in the following link. We'd like to express our sincere gratitude to members of AI4Design team at Google DeepMind for their insightful suggestions and support. Zhang and Shi were supported in part by National Science Foundation CAREER under Award #2427478, and by National Science Foundation and the Institute of Education Sciences, U.S. Department of Education under Award #2229873 - National AI Institute for Exceptional Education.

## Footnotes

*Work done during a part-time internship at Google DeepMind.

†Work done while at Google Research.

## References

[1] Namhyuk Ahn, Junsoo Lee, Chunggi Lee, Kunhee Kim, Daesik Kim, Seung-Hun Nam, and Kibeom Hong. Dreamstyler: Paint by style inversion with text-to-image diffusion models. In *Proceedings of the AAAI Conference on Artificial Intelligence*, volume 38, pages 674–681, 2024. 3, 7

[2] Tim Brooks, Aleksander Holynski, and Alexei A Efros. Instructpix2pix: Learning to follow image editing instructions. In *Proceedings of the IEEE/CVF Conference on Computer Vision and Pattern Recognition*, pages 18392–18402, 2023. 3, 19

[3] Huiwen Chang, Han Zhang, Lu Jiang, Ce Liu, and William T Freeman. Maskgit: Masked generative image transformer. In *Proceedings of the IEEE/CVF Conference on Computer Vision and Pattern Recognition*, pages 11315–11325, 2022. 3, 4

[4] Huiwen Chang, Han Zhang, Jarred Barber, AJ Maschinot, Jose Lezama, Lu Jiang, Ming-Hsuan Yang, Kevin Murphy, William T Freeman, Michael Rubinstein, et al. Muse: Text-to-image generation via masked generative transformers. *arXiv preprint arXiv:2301.00704*, 2023. 2, 3, 4

[5] Hila Chefer, Yuval Alaluf, Yael Vinker, Lior Wolf, and Daniel Cohen-Or. Attend-and-excite: Attention-based semantic guidance for text-to-image diffusion models. *ACM Transactions on Graphics (TOG)*, 42 (4):1–10, 2023. 3

[6] cloneofsimo. Low-rank adaptation for fast text-to-image diffusion fine-tuning, 2022. URL https://github.com/cloneofsimo/lora. 3, 6

[7] Patrick Esser, Robin Rombach, and Bjorn Ommer. Taming transformers for high-resolution image synthesis. In *Proceedings of the IEEE/CVF conference on computer vision and pattern recognition*, pages 12873–12883, 2021. 3

[8] FireFly. Firefly, 2022. URL https://www.adobe.com/products/firefly.html. 2

[9] Yarden Frenkel, Yael Vinker, Ariel Shamir, and Daniel Cohen-Or. Implicit style-content separation using b-lora. *arXiv preprint arXiv:2403.14572*, 2024. 6

[10] Rinon Gal, Yuval Alaluf, Yuval Atzmon, Or Patashnik, Amit H Bermano, Gal Chechik, and Daniel Cohen-Or. An image is worth one word: Personalizing text-to-image generation using textual inversion. *arXiv preprint arXiv:2208.01618*, 2022. 2, 3, 6

[11] Gemini. Gemini, 2024. URL https://gemini.google.com/. 2, 19

[12] Vidit Goel, Elia Peruzzo, Yifan Jiang, Dejia Xu, Xingqian Xu, Nicu Sebe, Trevor Darrell, Zhangyang Wang, and Humphrey Shi. Pair diffusion: A comprehensive multimodal object-level image editor. In *Proceedings of the IEEE/CVF Conference on Computer Vision and Pattern Recognition*, pages 8609–8618, 2024. 3

[13] Amir Hertz, Ron Mokady, Jay Tenenbaum, Kfir Aberman, Yael Pritch, and Daniel Cohen-Or. Prompt-to-prompt image editing with cross attention control. *arXiv preprint arXiv:2208.01626*, 2022. 3

[14] Amir Hertz, Andrey Voynov, Shlomi Fruchter, and Daniel Cohen-Or. Style aligned image generation via shared attention. *arXiv preprint arXiv:2312.02133*, 2023. 3

[15] Amir Hertz, Andrey Voynov, Shlomi Fruchter, and Daniel Cohen-Or. Style aligned image generation via shared attention. In *Proceedings of the IEEE/CVF Conference on Computer Vision and Pattern Recognition*, pages 4775–4785, 2024. 7

[16] Jonathan Ho and Tim Salimans. Classifier-free diffusion guidance. *arXiv preprint arXiv:2207.12598*, 2022. 4

[17] Jonathan Ho, Ajay Jain, and Pieter Abbeel. Denoising diffusion probabilistic models. *Advances in neural information processing systems*, 33:6840–6851, 2020. 3

[18] Neil Houlsby, Andrei Giurgiu, Stanislaw Jastrzebski, Bruna Morrone, Quentin De Laroussilhe, Andrea Gesmundo, Mona Attariyan, and Sylvain Gelly. Parameter-efficient transfer learning for nlp. In *International conference on machine learning*, pages 2790–2799. PMLR, 2019. 2, 3, 4

[19] Chih-Chung Hsu, Yi-Xiu Zhuang, and Chia-Yen Lee. Deep fake image detection based on pairwise learning. *Applied Sciences*, 10(1):370, 2020. 19

[20] Edward J Hu, Yelong Shen, Phillip Wallis, Zeyuan Allen-Zhu, Yuanzhi Li, Shean Wang, Lu Wang, and Weizhu Chen. Lora: Low-rank adaptation of large language models. *arXiv preprint arXiv:2106.09685*, 2021. 3, 6

[21] Tero Karras, Samuli Laine, and Timo Aila. A style-based generator architecture for generative adversarial networks. In *Proceedings of the IEEE/CVF conference on computer vision and pattern recognition*, pages 4401–4410, 2019. 3

[22] Bahjat Kawar, Shiran Zada, Oran Lang, Omer Tov, Huiwen Chang, Tali Dekel, Inbar Mosseri, and Michal Irani. Imagic: Text-based real image editing with diffusion models. In *Proceedings of the IEEE/CVF Conference on Computer Vision and Pattern Recognition*, pages 6007–6017, 2023. 3

[23] Diederik P Kingma and Jimmy Ba. Adam: A method for stochastic optimization. *arXiv preprint arXiv:1412.6980*, 2014. 15

[24] Hyung-Kwon Ko, Gwanmo Park, Hyeon Jeon, Jaemin Jo, Juho Kim, and Jinwook Seo. Large-scale text-to-image generation models for visual artists' creative works. In *Proceedings of the 28th international conference on intelligent user interfaces*, pages 919–933, 2023. 19

[25] Kyungmin Lee, Sangkyung Kwak, Kihyuk Sohn, and Jinwoo Shin. Direct consistency optimization for compositional text-to-image personalization. *arXiv preprint arXiv:2402.12004*, 2024. 5

[26] Brian Lester, Rami Al-Rfou, and Noah Constant. The power of scale for parameter-efficient prompt tuning. *arXiv preprint arXiv:2104.08691*, 2021. 3

[27] Haoming Lu, Hazarapet Tunanyan, Kai Wang, Shant Navasardyan, Zhangyang Wang, and Humphrey Shi. Specialist diffusion: Plug-and-play sample-efficient fine-tuning of text-to-image diffusion models to learn any unseen style. In *Proceedings of the IEEE/CVF Conference on Computer Vision and Pattern Recognition*, pages 14267–14276, 2023. 2

[28] Midjourney. Midjourney, 2022. URL https://www.midjourney.com. 2

[29] Eric Mitchell, Yoonho Lee, Alexander Khazatsky, Christopher D Manning, and Chelsea Finn. Detectgpt: Zero-shot machine-generated text detection using probability curvature. In *International Conference on Machine Learning*, pages 24950–24962. PMLR, 2023. 19

[30] Chong Mou, Xintao Wang, Liangbin Xie, Yanze Wu, Jian Zhang, Zhongang Qi, and Ying Shan. T2i-adapter: Learning adapters to dig out more controllable ability for text-to-image diffusion models. In *Proceedings of the AAAI Conference on Artificial Intelligence*, volume 38, pages 4296–4304, 2024. 3, 6

[31] Alex Nichol, Prafulla Dhariwal, Aditya Ramesh, Pranav Shyam, Pamela Mishkin, Bob McGrew, Ilya Sutskever, and Mark Chen. Glide: Towards photorealistic image generation and editing with text-guided diffusion models. *arXiv preprint arXiv:2112.10741*, 2021. 3, 4

[32] Or Patashnik, Daniel Garibi, Idan Azuri, Hadar Averbuch-Elor, and Daniel Cohen-Or. Localizing object-level shape variations with text-to-image diffusion models. In *Proceedings of the IEEE/CVF International Conference on Computer Vision*, pages 23051–23061, 2023. 3

[33] Sayak Paul Pedro Cuenca. Using lora for efficient stable diffusion fine-tuning, 2023. URL https://huggingface.co/blog/lora. 6

[34] Picsart. Picsart, 2021. URL https://picsart.com/. 2

[35] Alec Radford, Jong Wook Kim, Chris Hallacy, Aditya Ramesh, Gabriel Goh, Sandhini Agarwal, Girish Sastry, Amanda Askell, Pamela Mishkin, Jack Clark, et al. Learning transferable visual models from natural language supervision. In *International conference on machine learning*, pages 8748–8763. PMLR, 2021. 3, 9

[36] Colin Raffel, Noam Shazeer, Adam Roberts, Katherine Lee, Sharan Narang, Michael Matena, Yanqi Zhou, Wei Li, and Peter J Liu. Exploring the limits of transfer learning with a unified text-to-text transformer. *Journal of machine learning research*, 21(140):1–67, 2020. 3

[37] Aditya Ramesh, Mikhail Pavlov, Gabriel Goh, Scott Gray, Chelsea Voss, Alec Radford, Mark Chen, and Ilya Sutskever. Zero-shot text-to-image generation. In *International conference on machine learning*, pages 8821–8831. Pmlr, 2021. 3

[38] Robin Rombach, Andreas Blattmann, Dominik Lorenz, Patrick Esser, and Björn Ommer. High-resolution image synthesis with latent diffusion models. In *Proceedings of the IEEE/CVF conference on computer vision and pattern recognition*, pages 10684–10695, 2022. 2

[39] Nataniel Ruiz, Yuanzhen Li, Varun Jampani, Yael Pritch, Michael Rubinstein, and Kfir Aberman. Dreambooth: Fine tuning text-to-image diffusion models for subject-driven generation. In *Proceedings of the IEEE/CVF Conference on Computer Vision and Pattern Recognition*, pages 22500–22510, 2023. 2, 3

[40] Chitwan Saharia, William Chan, Saurabh Saxena, Lala Li, Jay Whang, Emily Denton, Seyed Kamyar Seyed Ghasemipour, Burcu Karagol Ayan, S Sara Mahdavi, Rapha Gontijo Lopes, et al. Photorealistic text-to-image diffusion models with deep language understanding. *arXiv preprint arXiv:2205.11487*, 2022. 2, 3, 4

[41] Kihyuk Sohn, Nataniel Ruiz, Kimin Lee, Daniel Castro Chin, Irina Blok, Huiwen Chang, Jarred Barber, Lu Jiang, Glenn Entis, Yuanzhen Li, et al. Styledrop: Text-to-image generation in any style. *arXiv preprint arXiv:2306.00983*, 2023. 2, 3, 4, 5, 6, 7, 10, 15

[42] Yang Song and Stefano Ermon. Generative modeling by estimating gradients of the data distribution. *Advances in neural information processing systems*, 32, 2019. 3

[43] Christopher Teo, Milad Abdollahzadeh, and Ngai-Man Man Cheung. On measuring fairness in generative models. *Advances in Neural Information Processing Systems*, 36, 2024. 19

[44] Su Wang, Chitwan Saharia, Ceslee Montgomery, Jordi Pont-Tuset, Shai Noy, Stefano Pellegrini, Yasumasa Onoe, Sarah Laszlo, David J Fleet, Radu Soricut, et al. Imagen editor and editbench: Advancing and evaluating text-guided image inpainting. In *Proceedings of the IEEE/CVF conference on computer vision and pattern recognition*, pages 18359–18369, 2023. 3

[45] Yuxin Wen, Neel Jain, John Kirchenbauer, Micah Goldblum, Jonas Geiping, and Tom Goldstein. Hard prompts made easy: Gradient-based discrete optimization for prompt tuning and discovery. *Advances in Neural Information Processing Systems*, 36, 2024. 2

[46] Xingqian Xu, Zhangyang Wang, Gong Zhang, Kai Wang, and Humphrey Shi. Versatile diffusion: Text, images and variations all in one diffusion model. In *Proceedings of the IEEE/CVF International Conference on Computer Vision*, pages 7754–7765, 2023. 3

[47] Xingqian Xu, Jiayi Guo, Zhangyang Wang, Gao Huang, Irfan Essa, and Humphrey Shi. Prompt-free diffusion: Taking" text" out of text-to-image diffusion models. In *Proceedings of the IEEE/CVF Conference on Computer Vision and Pattern Recognition*, pages 8682–8692, 2024. 3

[48] Hu Ye, Jun Zhang, Sibo Liu, Xiao Han, and Wei Yang. Ip-adapter: Text compatible image prompt adapter for text-to-image diffusion models. *arXiv preprint arXiv:2308.06721*, 2023. 2, 3, 7

[49] Jiahui Yu, Xin Li, Jing Yu Koh, Han Zhang, Ruoming Pang, James Qin, Alexander Ku, Yuanzhong Xu, Jason Baldridge, and Yonghui Wu. Vector-quantized image modeling with improved vqgan. *arXiv preprint arXiv:2110.04627*, 2021. 3

[50] Jiahui Yu, Yuanzhong Xu, Jing Yu Koh, Thang Luong, Gunjan Baid, Zirui Wang, Vijay Vasudevan, Alexander Ku, Yinfei Yang, Burcu Karagol Ayan, et al. Scaling autoregressive models for content-rich text-to-image generation. *arXiv preprint arXiv:2206.10789*, 2(3):5, 2022. 3, 4, 8

[51] Eric Zhang, Kai Wang, Xingqian Xu, Zhangyang Wang, and Humphrey Shi. Forget-me-not: Learning to forget in text-to-image diffusion models. *arXiv preprint arXiv:2303.17591*, 2023. 3

[52] Lvmin Zhang, Anyi Rao, and Maneesh Agrawala. Adding conditional control to text-to-image diffusion models. In *Proceedings of the IEEE/CVF International Conference on Computer Vision*, pages 3836–3847, 2023. 2, 3

# A Appendix

## A.1 Style Images and Prompts

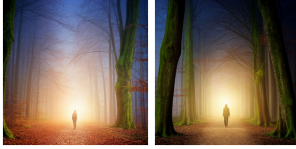
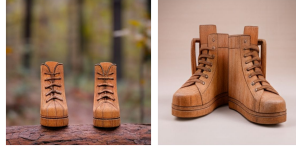
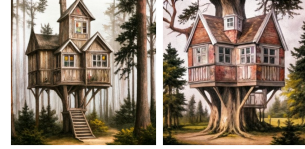

A person walking on the road with {fallen leaves|~~fallen leaves~~} in beautifully lit mythical photograph. there are big trees by the road.

Sneakers in matte and worn out textured wooden sculpture {in the forest|~~in the forest~~}.

a tree house made of {woods|bricks} in watercolor painting style.

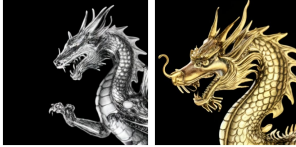
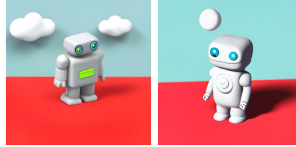
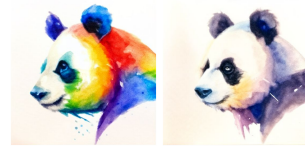

a dragon in glossy {silvery|golden} metal sculpture on a solid black background.

A robot in glossy textured 3d rendering style on a light red ground. there are {clouds|moon} in a light blue sky.

A panda in {rainbow color|black and white} watercolor painting style on a white background.

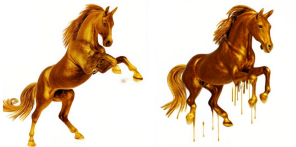
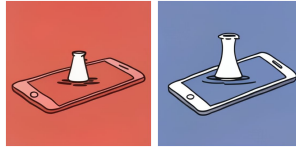
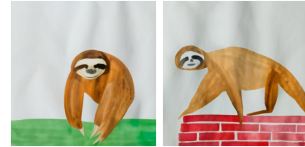

A horse {~~with drips~~|with drips} in melting golden 3d rendering style on a solid white background.

A vase drowning into the phone cartoon line drawing style on a {red|blue} background.

A sloth on the {grass|bricks} in watercolor painting style on a white background.

Figure 9: Full text prompts used for image generation in Fig. 1.

## A.2 Styles for Quantitative Evaluation

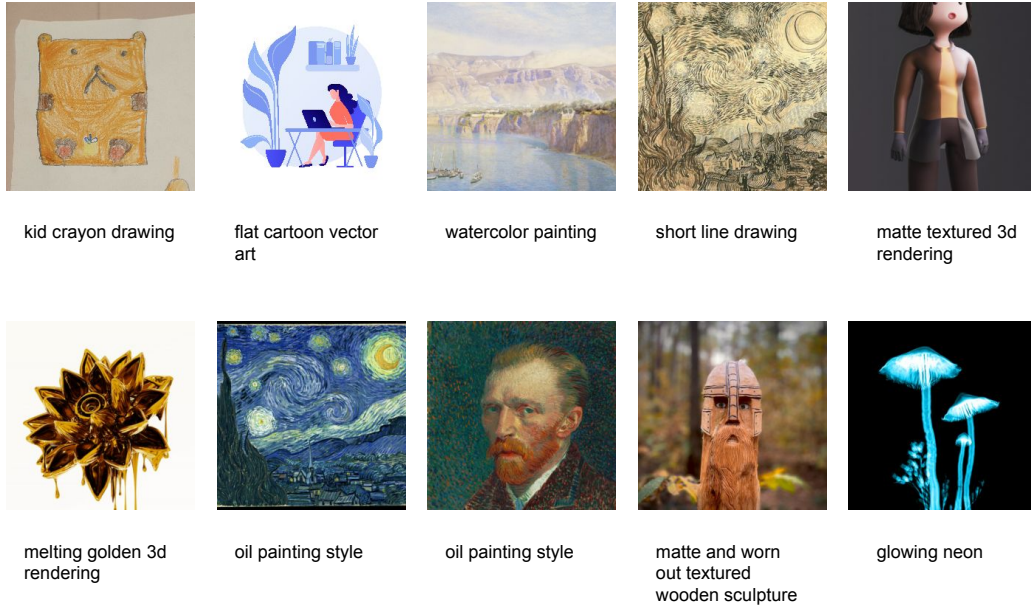

| | | | | |
|---|---|---|---|---|
| kid crayon drawing | flat cartoon vector art | watercolor painting | short line drawing | matte textured 3d rendering |
| melting golden 3d rendering | oil painting style | oil painting style | matte and worn out textured wooden sculpture | glowing neon |

Figure 10: Styles used for quantitative evaluation.

## A.3 Implementation Details

We train FineStlye using Adam optimizer [23] on TPUv4 with a batch size of 8. See Tab. 3 for detailed hyperparamters.

Table 3: Hyperparameters for optimizer, adapter architecture, and synthesis.

| | FineStyle | StyleDrop |
|---|---|---|
| Learning rate | 0.00003 | 0.00003 |
| Batch size | 8 | 8 |
| # steps | 2500 | 1000 |
| d_prj | 4 | 1 |
| is_shared | True | False |
| # adapter parameters | 0.32M | 0.17M |
| # decoding step | 64 | 64 |
| temperature | 4.5 | 4.5 |
| $\lambda_1$ | 5.0 | 5.0 |
| $\lambda_2$ | 5.0 | 0.0 |
| $\lambda_{muse}$ | 0.0 | 5.0 |

StyeDrop uses classifer-free guidance with Muse model instead of fine-tuned model as in Eq. (5). For more details, please see [41].

$$v_k = \hat{\texttt{G}}(v_{k-1}, \texttt{T}(t)) + \lambda_1(\hat{\texttt{G}}(v_{k-1}, \texttt{T}(t)) - \texttt{G}(v_{k-1}, \texttt{T}(t))) + \lambda_{muse}(\texttt{G}(v_{k-1}, \texttt{T}(t)) - \texttt{G}(v_{k-1}, \texttt{T}(n)))$$
$$(5)$$

## A.4 Human Evaluation



## Instructions

A reference style is displayed in the photo provided below.

We then show you two machine generated images.

1. Select the image that best corresponds to the style image.
2. Select the image that best corresponds to the prompt.
3. Select the image that best corresponds to common sense and is structurally sound.

*Review this definition of style:*
*Style (from Merriam-Webster): A particular manner or technique by which something is done, created or performed.*

*How to decide if the image follows the prompt:*
*1. All elements in the prompt should be present in the image*
*2. Additional elements not in the prompt should not be generated in the image and if they are they should be considered hallucinations*

**Reference Style Image**



Given the reference image above and these two machine-generated output images, select which machine-generated output better matches the **style** of the reference image *



○ Image A
○ Image B
○ Cannot Determine / Both Equally

**Reference prompt**: "a ceiling fan, indoor scenes, in watercolor painting style. there * is a mountain in the background."

Select which machine-generated output images better matches the **reference prompt** and without regard for the reference style image.

*How to decide if the image follows the prompt:*
*1. All elements in the prompt should be present in the generated image*
*2. Elements that are not in the prompt should not be present in the image (example: if the image does not say "buildings in the background" they should not be present)*



○ Image A
○ Image B
○ Cannot Determine / Both Equally

**Reference prompt**: "a ceiling fan, indoor scenes, in watercolor painting style. there * is a mountain in the background."

Select which image is makes more common sense and is structurally sound. (Example: a generated image of a horse should have 4 legs unless otherwise specified by the prompt)



○ Image A
○ Image B
○ Cannot Determine / Both Equally

Figure 11: The interface of human evaluation form.

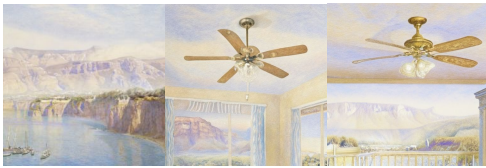

a ceiling fan, indoor scenes, in watercolor
painting style. there is a mountain in the
background.

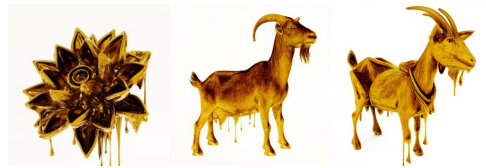

a goat with drips in melting golden 3d
rendering style on a solid white background.

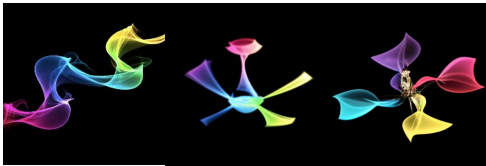

a ceiling fan, indoor scenes, in rainbow
colored flowing design on a solid black
background.

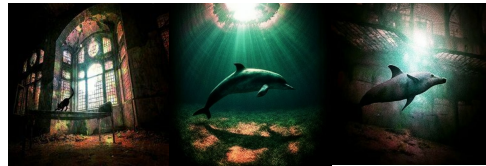

a dolphin, animals, in well lit haunted photograph.

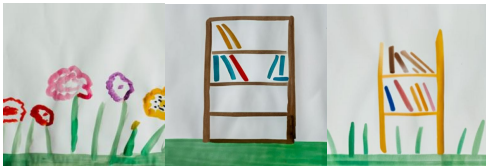

a bookshelf on the grass in watercolor painting
style on a white background.

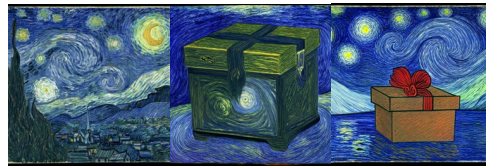

a box in oil painting style on a blue
background.

Figure 12: Examples of human evaluation triplets. In a triplet, from left to right, they are the reference, FineStyle synthesis, and StyleDrop synthesis.

### A.5 Derivation of Concept Attention Map

```python
import jax.numpy as jnp

def aggregate_xattn_by_phrase(
    xattn_matrix: jnp.ndarray,
    prompt_tokens: List[str],
    phrase_tokens: List[str],
):
  """
  aggregate xattn maps of a phrase in a prompt across all layers,
    heads of a generative transformer.
  Args:
    xattn_matrix: a tensor of cross-attention weights, num_layer x
    batch x num_head x visual_token_len x text_token_len
    prompt_tokens: a list of tokenized prompt
    phrase_tokens: a list of tokenized phrase

  Return:
    a tensor, batch x visual_token_len
  """
  matched = []
  for i in range(len(prompt_tokens)):
    if (
        prompt_tokens[i] == phrase_tokens[0]
        and prompt_tokens[i : i + len(phrase_tokens)] == phrase_tokens
    ):
      matched.append((i, i + len(phrase_tokens)))

  xattn_of_phrase = []
  for start, end in matched:
    xattn_of_phrase.append(xattn_matrix[..., start:end])
  return jnp.stack(xattn_of_phrase).mean(axis=[0, 1, 3, 5])

# example prompt and phrase tokens
prompt_tokens = ["a", "cat", "is", "running", "."]
phrase_tokens = ["a", "cat"]
```

### A.6 Automate Prompt Generation with vLLM

Input to LLM: "`Give a brief description of the contents image`
`and the style of the image. List the main 3 objects or`
`entities in the image. Take the style description and write`
`it combined with each of the 3 entities listed, for example` 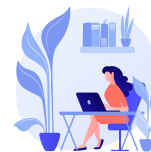
`'dog in a watercolor painting style'`"

Output: "`A woman sits at a desk working on her laptop, surrounded by`
`houseplants and a small bookshelf. The image is in a flat, vector style`
`with a blue and purple color scheme. The image is a digital illustration of`
`a woman working from home.`

`* **Woman:** Woman in a flat vector graphics style with a blue and purple`
`palette.`
`* **Laptop:** Laptop in a flat vector graphics style with a blue and purple`
`palette.`
`* **Plants:** Plants in a flat vector graphics style with a blue and purple`
`palette. `"

Figure 13: An example of using vLLM Gemini to automate the process of writing caption and sub-prompts for a style image.

We recognize the cumbersome nature of requiring users to manually identify specific concepts of interest and craft a detailed caption for a reference image. To address this, we explore the potential of utilizing state-of-the-art vision large language models (vLLMs) to automate this process. Fig. 13 shows that we successfully prompt Gemini [11] with a reference image to get a comprehensive caption and 3 concept-oriented sub-prompts.

### A.7    Limitations

The diversity of fine-grained styles extends far beyond the evaluation set used in this paper. While our method successfully achieves fine-grained control over properties such as color, texture, and foreground objects, it does not imply that it can manage every possible style composition defined by a text prompt. The exploration of extending the scope of fine-grained style controllability will be addressed in future work.

### A.8    Broader Impact

As shown in Fig. 1, FineStyle enables people to create unique artworks in their own style assets with more controllability. This can benefit both art designers and general users by enhancing their work productivity and adding enjoyment to their lives [2, 24]. However it is necessary to be aware of rare but negative potentials of image generation. These include generating sensitive content (*e.g.*, Deepfake [19]) or generating images that violate copyright. Clear steps have been outlined to prevent this [43]. Steps include carefully curating data for training such as not to include inappropriate contents, and then thoroughly analyzing the delta between the model's generated data and real data and develop a detection framework [29].

